# SINGLE NEURON MODEL: RESPONSE TO WEAK MODULATION IN THE PRESENCE OF NOISE

*A. R. Bulsara and E. W. Jacobs*

Naval Ocean Systems Center, Materials Research Branch, San Diego, CA 92129

*F. Moss*

Physics Dept., Univ. of Missouri, St. Louis, MO 63121

### ABSTRACT

We consider a noisy bistable single neuron model driven by a periodic external modulation. The modulation introduces a correlated switching between states driven by the noise. The information flow through the system from the modulation to the output switching events, leads to a succession of strong peaks in the power spectrum. The signal-to-noise ratio (SNR) obtained from this power spectrum is a measure of the information content in the neuron response. With increasing noise intensity, the SNR passes through a maximum, an effect which has been called *stochastic resonance*. We treat the problem within the framework of a recently developed approximate theory, valid in the limits of weak noise intensity, weak periodic forcing and low forcing frequency. A comparison of the results of this theory with those obtained from a linear system FFT is also presented.

## INTRODUCTION

Recently, there has been an upsurge of interest in *single* or few-neuron nonlinear dynamics (see e.g. Li and Hopfield, 1989; Tuckwell, 1988; Paulus, Gass and Mandell, 1990; Aihara, Takake and Toyoda, 1990). However, the precise relationship between the many-neuron connected model and a single effective neuron dynamics has not been examined in detail. Schieve, Bulsara and Davis (1991) have considered a network of N symmetrically interconnected neurons embodied, for example in the "connectionist" models of Hopfield (1982, 1984) or Shamma (1989) (the latter corresponding to a mammalian auditory network). Through an adiabatic elimination procedure, they have obtained, in closed form, the dynamics of a single neuron from the system of coupled differential equations describing the N-neuron problem. The problem has been treated both deterministically and stochastically (through the inclusion of additive and multiplicative noise terms). It is important to point out that the work of Schieve, Bulsara, and Davis does not include *a priori* a self-coupling term, although the inclusion of such a term can be readily implemented in their theory; this has been done by Bulsara and Schieve (1991). Rather, their theory results in an explicit form of the self-coupling term, in terms of the parameters of the remaining neurons in the network. This term, in effect, renormalizes the self-coupling term in the Shamma and Hopfield models. The reduced or "effective" neuron model is expected to reproduce some of the gross features of biological neurons. The fact that simple single neuron models, such as the model to be considered in this work, can indeed reproduce several features observed in biological experiments has been strikingly demonstrated by Longtin, Bulsara and Moss (1991) through their construction of the inter-spike-interval histograms (ISIHs) using a Schmidt trigger to model the neuron. The results of their simple model agree remarkably well with data obtained in two different experiments (on the auditory nerve fiber of squirrel monkey (Rose, Brugge, Andersen and Hind, 1967) and on the cat visual cortex (Siegal, 1990)).

In this work, we consider such a "reduced" neural element subject to a weak periodic external modulation. The modulation introduces a correlated switching between the

bistable states, driven by the noise with the signal-to-noise ratio (SNR) obtained from the power spectrum, being taken as a measure of the information content in the neuron response. As the additive noise variance increases, the SNR passes through a maximum. This effect has been called "stochastic resonance" and describes a phenomenon in which the noise actually enhances the information content, i.e., the observability of the signal. Stochastic resonance has been observed in a modulated ring laser experiment (McNamara, Wiesenfeld and Roy, 1988; Vemuri and Roy, 1989) as well as in electron paramagnetic resonance experiments (Gammaitoni, Martinelli, Pardi and Santucci, 1991) and in a modulated magnetoselastic ribbon (Spano and Ditto, 1991). The introduction of multiplicative noise (in the coefficient of the sigmoid transfer function) tends to degrade this effect.

## THE MODEL; STOCHASTIC RESONANCE

The reduced neuron model consists of a single Hopfield-type computational element, which may be modeled as a R-C circuit with nonlinear feedback provided by an operational amplifier having a sigmoid transfer function. The equation (which may be rigorously derived from a fully connected network model as outlined in the preceding section) may be cast in the form,

$$\dot{x} + a\,x - b\,\tanh x = x_0 + F(t)\,,\tag{1}$$

where $F(t)$ is Gaussian delta-correlated noise with zero mean and variance $2D$, $x_0$ being a dc input (which we set equal to zero for the remainder of this work). An analysis of (1), including multiplicative noise effects, has been given by Bulsara, Boss and Jacobs (1989). For the purposes of the current work, we note that the neuron may be treated as a particle in a one-dimensional potential given by,

$$U(x) = \frac{a\,x^2}{2} - b\,\ln\cosh x\,,\tag{2}$$

$x$ being the one-dimensional state variable representing the membrane potential. In general, the coefficients $a$ and $b$ depend on the details of the interaction of our reference neuron to the remaining neurons in the network (Schieve, Bulsara and Davis, 1990). The potential described by (2) is bimodal for $\eta > 1$ with the extrema occurring at (we set $a=1$ throughout the remainder of this work),

$$c = 0,\quad \pm\left[1 - \frac{1-\tanh b}{1 - b\,\mathrm{sech}^2 b}\right] \approx b\,\tanh b\,,\tag{3}$$

the approximation holding for large $b$. Note that the N-shaped characteristic inherent in the firing dynamics derived from the Hodgkin-Huxley equations (Rinzel and Ermentrout, 1990) is markedly similar to the plot of $dU/dx$ vs. $x$ for the simple bistable system (1). For a stationary potential, and for $D \ll U_0$ where $U_0$ is the depth of the deterministic potential, the probability that a switching event will occur in unit time, i.e. the switching rate, is given by the Kramers frequency (Kramers, 1940),

$$r_0 = \left\{ D \int_{-c}^{0} dy\,\exp\left(U(y)/D\right) \int_{-\infty}^{y} dz\,\exp\left(-U(z)/D\right) \right\}^{-1},\tag{4a}$$

which, for small noise, may be cast in the form (the local equilibrium assumption of Kramers),

$$r_0 \approx (2\pi)^{-1}\left[\,|\,U^{(2)}(0)\,|\,U^{(2)}(c)\,\right]^{1/2}\exp\left(-U_0/D\right),\tag{4b}$$

where $U^{(2)}(x) \equiv d^2U/dx^2$.

We now include a periodic modulation term $\epsilon\sin\omega t$ on the right-hand-side of (1) (note that for $\epsilon < 2(b-1)^3/(3b)$ one does not observe switching in the noise-free system). This leads to a modulation (i.e. rocking) of the potential (2) with time: an additional term $-x\epsilon\sin\omega t$ is now present on the right-hand-side of (2). In this case, the Kramers rate (4) becomes time-dependent:

$$r(t) \approx r_0\exp(-x\epsilon\sin\omega t/D)\,,\tag{5}$$

which is accurate only for $\epsilon \ll U_0$ and $\omega \ll \{U^{(2)}(\pm c)\}^{1/2}$. The latter condition is referred to as the *adiabatic approximation*. It ensures that the probability density corresponding to

the time-modulated potential is approximately stationary (the modulation is slow enough that the instantaneous probability density can "adiabatically" relax to a succession of quasi-stationary states).

We now follow the work of McNamara and Wiesenfeld (1989), developing a two-state model by introducing a probability of finding the system in the left or right well of the potential. A rate equation is constructed based on the Kramers rate $r(t)$ given by (5). Within the framework of the adiabatic approximation, this rate equation may be integrated to yield the time-dependent conditional probability density function for finding the system in a given well of the potential. This leads directly to the autocorrelation function $<z(t)\,z(t+\tau)>$ and finally, via the Wiener-Khinchine theorem, to the power spectral density $P(\Omega)$. The details are given by Bulsara, Jacobs, Zhou, Moss and Kiss (1991):

$$P(\Omega) = \left[1 - \frac{2r_0^2\epsilon^2c^2}{D^2(4r_0^2 + \Omega^2)}\right]\left[\frac{8c^2r_0}{4r_0^2 + \Omega^2}\right] + \frac{4\pi c^4 r_0^2 \epsilon^2}{D^2(4r_0^2 + \Omega^2)}\,\delta(\omega - \Omega)\,, \tag{6}$$

where the first term on the right-hand-side represents the noise background, the second term being the signal strength. Taking into account the finite bandwidth of the measuring system, we replace (for the purpose of comparison with experimental results) the delta-function in (6) by the quantity $(\Delta\omega)^{-1}$ where $\Delta\omega$ is the width of a frequency bin in the (experimental) Fourier transformation. We introduce signal-to-noise ratio $SNR = 10\log R$ in decibels, where $R$ is given by

$$R \equiv 1 + \frac{4\pi c^4 r_0^2 \epsilon^2}{D^2(4r_0^2 + \omega^2)}(\Delta\omega)^{-1}\left[1 - \frac{2r_0^2\epsilon^2c^2}{D^2(4r_0^2 + \omega^2)}\right]^{-1}\left[\frac{4r_0^2 + \omega^2}{8c^2r_0}\right]. \tag{7}$$

In writing down the above expressions, the approximate Kramers rate (4b) has been used. However, in what follows, we discuss the effects of replacing it by the exact expression (4a). The location of the maximum of the SNR is found by differentiating the above equation; it depends on the amplitude $\epsilon$ and the frequency $\omega$ of the modulation, as well as the additive noise variance $D$ and the parameters $a$ and $b$ in the potential.

The SNR computed via the above expression increases as the modulation frequency is lowered relative to the Kramers frequency. Lowering the modulation frequency also sharpens the resonance peak, and shifts it to lower noise values, an effect that has been demonstrated, for example, by Bulsara, Jacobs, Zhou, Moss and Kiss (1991). The above may be readily explained. The effect of the weak modulating signal is to alternately raise and lower the potential well with respect to the barrier height $U_0$. In the absence of noise and for $\epsilon \ll U_0$, the system cannot switch states, i.e. no information is transferred to the output. In the presence of noise, however, the system can switch states through stochastic activation over the barrier. Although the switching process is statistical, the transition probability is periodically modulated by the external signal. Hence, the output will be correlated, to some degree, with the input signal (the modulation "clocks" the escape events and the whole process will be optimized if the noise by itself produces, on average, two escapes within one modulation cycle).

Figure 1 shows the SNR as a function of the noise variance $2D$. The potential barrier height $U_0 = 2.4$ for the $b = 2.5$ case considered. Curves corresponding to the adiabatic expression (7), as well as the SNR obtained through an exact (numerical) calculation of the Kramers rate, using (4a) are shown, along with the data points obtained via direct numerical simulation of (1). The Kramers rate at the maximum $(2D \approx U_0)$ of the SNR curve is 0.72. This is much greater than the driving frequency $\omega = 0.0393$ used in this plot. The curve computed using the exact expression (4a) fits the numerically obtained data points better than the adiabatic curve at high noise strengths. This is to be expected in light of the approximations used in deriving (4b) from (4a). Also, the expression (6) has been derived from a two-state theory (taking no account of the potential). At low noise, we expect the two-state theory to agree with the actual system more closely. This is reflected in the resonance curves of figure 1 with the adiabatic curve differing (at the maximum) from the data points by approximately 1db. We reiterate that the SNR, as well as the agreement between the data points and the theoretical curves improves as the modulation frequency is lowered relative to the Kramers rate (for a fixed frequency this can be achieved by changing the potential barrier height via the parameters $a$ and $b$ in (2)). On the same plot, we show the SNR obtained by computing directly the Fourier transform of the signal and noise. At very

low noise, the "ideal linear filter" yields results that are considerably better than stochastic resonance. However, at moderate-to-high noise, the stochastic resonance, which may be looked upon as a "nonlinear filter", offers at least a 2.5db improvement for the parameters of the figure. As indicated above, the improvement in performance achieved by stochastic resonance over the "ideal linear filter" may be enhanced by raising the Kramers frequency of the nonlinear filter relative to the modulation frequency $\omega$. In fact, as long as the basic conditions of stochastic resonance are realized, the nonlinear filter will outperform the best linear filter except at very low noise.

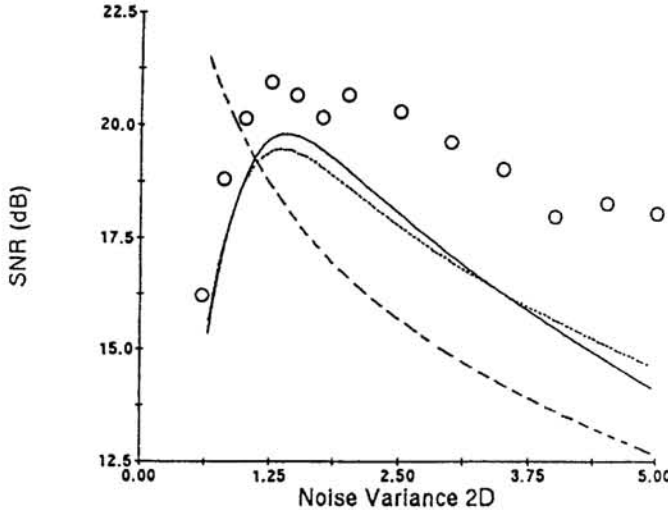

Fig 1. SNR using adiabatic theory, eqn. (7), with $(b,\omega,\epsilon)=$ (2.5,0.0393,0.3) and $r_0$ given by (4b) (solid curve) and (4a) (dotted curve). Data points correspond to SNR obtained via direct simulation of (1) (frequency resolution $=6.1\times10^{-5}$ Hz). Dashed curve corresponds to best possible linear filter (see text).

## Multiplicative Noise Effects

We now consider the case when the neuron is exposed to both additive and multiplicative noise. In this case, we set $b(t) = b_0 + \xi(t)$ where

$$<\xi(t)> = 0, \quad <\xi(t)\,\xi(s)> = 2D_m\,\delta(t-s)\,. \tag{8}$$

In a real system such fluctuations might arise through the interaction of the neuron with other neurons in the network or with external fluctuations. In fact, Schieve, Bulsara and Davis (1991) have shown that when one derives the "reduced" neuron dynamics in the form (1) from a fully connected N-neuron network with fluctuating synaptic couplings, then the resulting dynamics contain multiplicative noise terms of the kind being discussed here. Even Langevin noise by itself can introduce a pitchfork bifurcation into the long-time dynamics of such a reduced neuron model under the appropriate conditions (Bulsara and Schieve, 1991). In an earlier publication (Bulsara, Boss and Jacobs, 1989), it was shown that these fluctuations can qualitatively alter the behavior of the stationary probability density function that describes the stochastic response of the neuron. In particular, the multiplicative noise may induce additional peaks or erase peaks already present in the density (see for example Horsthemke and Lefever 1984). In this work we maintain $D_m$ sufficiently small that such effects are absent.

In the absence of modulation, one can write down a Fokker Planck equation for the probability density function $p(x,t)$ describing the neuron response:

$$\frac{\partial p}{\partial t} = -\frac{\partial}{\partial x}\left[\alpha(x)p\right] + \frac{1}{2}\frac{\partial^2}{\partial x^2}\left[\beta(x)p\right], \tag{9}$$

where

$$\alpha(x) \equiv -x + b_0\tanh x + D_m\tanh x\,\mathrm{sech}^2 x\,,$$
$$\beta(x) \equiv 2(D + D_m\tanh^2 x)\,, \tag{10}$$

$D$ being the additive noise intensity. In the steady state, (9) may be solved to yield a "macroscopic potential" function analogous to the function $U(x)$ defined in (2):

$$U(x) = -2\int^x \frac{\alpha(z)}{\beta(z)}\,dz + \ln\beta(x)\,. \tag{11}$$

From (11), one obtains the turning points of the potential through the solution of the transcendental equation

$$z - b_0 \tanh z + D_m \tanh z \operatorname{sech}^2 z = 0 . \tag{12}$$

The modified Kramers rate, $r_{0m}$, for this $z$-dependent diffusion process has been derived by Englund, Snapp and Schieve (1984):

$$r_{0m} = \frac{\beta(0)}{2\pi} |\, U^{(2)}(z_1) |\, U^{(2)}(0)\, |\, |^{1/2} \exp |\, U(z_1) - U(0) |, \tag{13}$$

where the maximum of the potential occurs at $z=0$ and the left minimum occurs at $z=z_1$.

If we now assume that a weak sinusoidal modulation $\epsilon \sin \omega t$ is present, we may once again introduce this term into the potential as in the preceding case, again making the adiabatic approximation. We easily obtain for the modified time-dependent Kramers rate,

$$r_\pm(t) = \frac{\beta(0)}{4\pi} |\, U^{(2)}(z_1) |\, U^{(2)}(0)\, |\, |^{1/2} \exp \left[ U(z_1) - U(0) \pm 2 \int_0^{z_1} \frac{\epsilon \sin \omega t}{\beta(z)} \, dz \right] . \tag{14}$$

Following the same procedure as we used in the additive noise case, we can obtain the ratio $R = 1 + S/\Delta \omega N$, for the case of both noises being present. The result is,

$$R = 1 + \pi \gamma_0 \eta_0^2 (\Delta \omega)^{-1} \left[ 1 - \frac{2\gamma_0^2 \eta_0^2}{\gamma_0^2 + \eta_0^2} \right]^{-1}, \tag{15}$$

where,

$$\gamma_0 \equiv \frac{\beta(0)}{2\pi} |\, U^{(2)}(z_1) |\, U^{(2)}(0)\, |\, |^{1/2} \exp |\, U(z_1) - U(0) |, \tag{16a}$$

and

$$\eta_0 \equiv \epsilon \int_0^{z_1} \frac{dz}{\beta(z)} = \frac{\epsilon}{2(D + D_m)} \left[ z_1 + m^{1/2} \tan^{-1}( m^{1/2} \tanh z_1) \right], \tag{16b}$$

with $m \equiv D_m/D$.

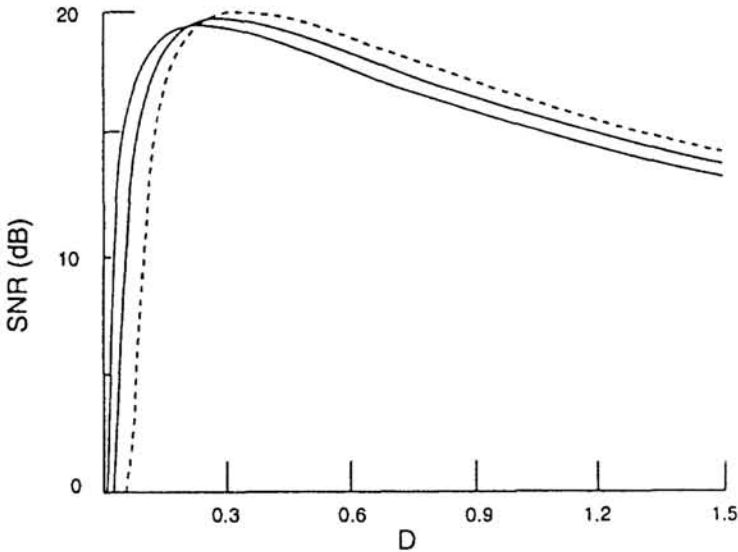

Fig 2. Effect of multiplicative noise, eqn. (15). $(b, \omega, \epsilon) = (2, 0.31, 0.4)$ and $D_m = 0$ (top curve), 0.1 (middle curve) and 0.2 (bottom curve).

In figure 2 we show the effects of both additive and multiplicative noise by plotting the SNR for a fixed external frequency $\omega = 0.31$ with $(b_0, \epsilon) = (2, 0.4)$ as a function of the additive noise intensity $D$. The curves correspond to different values of $D_m$, with the uppermost curve corresponding to $D_m = 0$, i.e., for the case of additive noise only. We note that increasing $D_m$ leads to a decrease in the SNR as well as a shift in its maximum to lower values of $D$. These effects are easily explained using the results of Bulsara, Boss and Jacobs

(1989), wherein it was shown that the effect of multiplicative noise is to decrease, on average, the potential barrier height and to shift the locations of the stable steady states. This leads to a degradation of the stochastic resonance effect at large $D_m$ while shifting the location of the maximum toward lower $D$.

## THE POWER SPECTRUM

We turn now to the power spectrum obtained via direct numerical simulation of the dynamics (1). It is evident that a time series obtained by numerical simulation of (1) would display switching events between the stable states of the potential, the residence time in each state being a random variable. The intrawell motion consists of a random component superimposed on a harmonic component, the latter increasing as the amplitude $\epsilon$ of the modulation increases. In the low noise limit, the deterministic motion dominates. However, the adiabatic theory used in deriving the expressions (6) and (7) is a two-state theory that simply follows the switching events between the states but takes no account of this intrawell motion. Accordingly, in what follows, we draw the distinction between the full dynamics obtained via direct simulation of (1) and the "equivalent two-state dynamics" obtained by passing the output through a two-state filter. Such a filter is realized digitally by replacing the time series obtained from a simulation of (1) with a time series wherein the $x$ variable takes on the values $x = \pm c$, depending on which state the system is in. Figure 3 shows the power spectral density obtained from this equivalent two-state system. The top curve represents the signal-free case and the bottom curve shows the effects of turning on the signal. Two features are readily apparent:

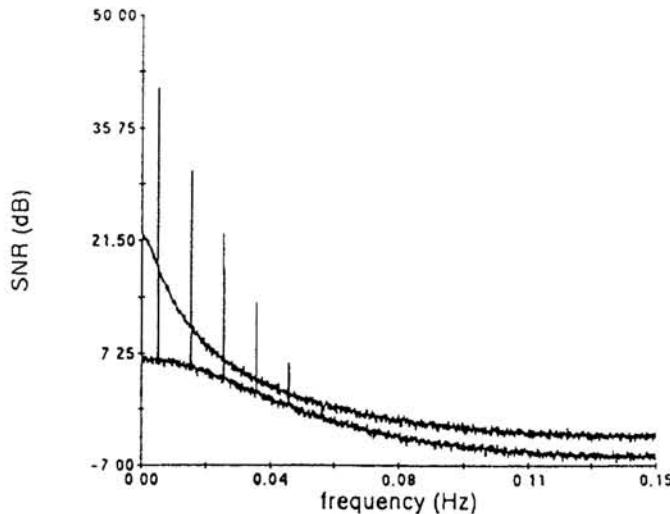

Fig 3. Power spectral density via direct simulation of (1).
$(b, \omega, \epsilon, 2D) = (1.6056, 0.03, 0.65, 0.25)$.
Bottom curve: $\epsilon=0$ case.

1. The power spectrum displays *odd* harmonics of the modulation; this is a hallmark of stochastic resonance (Zhou and Moss, 1990). If one destroys the symmetry of the potential (1) (through the introduction of a small dc driving term, for example), the even harmonics of the modulation appear.
2. The noise floor is lowered when the signal is turned on. This effect is particularly striking in the two-state dynamics. It stems from the fact that the total area under the spectral density curves in figure 3 (i.e. the total power) must be conserved (a consequence of Parseval's theorem). *The power in the signal spikes therefore grows at the expense of the background noise* power. This is a unique feature of weakly modulated bistable noisy systems of the type under consideration in this work, and graphically illustrates the ability of noise to assist information flow to the output (the signal). The effect may be quantified on examining equation (6) above. The noise power spectral density (represented by the first term on the right-hand-side) decreases as the term $2r_0^2 \epsilon^2 c^2 \{D^2(4r_0^2 + \Omega^2)\}^{-1}$ approaches unity. This reduction in the noise floor is most pronounced when the signal is of low frequency (compared to the Kramers rate) and large amplitude. A similar, effect may be observed in the spectral density corresponding to the full system dynamics. In this case, the total power is only approximately conserved (in a finite bandwidth) and the effect is not so

pronounced.

## DISCUSSION

In this paper we have presented the details of a cooperative stochastic process that occurs in *nonlinear* systems subject to weak deterministic modulating signals embedded in a white noise background. The so-called "stochastic resonance" phenomenon may actually be interpreted as a noise-assisted flow of information to the output. The fact that such simple nonlinear dynamic systems (e.g. an electronic Schmidt trigger) are readily realizeable in hardware, points to the possible utility of this technique (far beyond the application to signal processing in simple neural networks) as a nonlinear filter. We have demonstrated that, by suitably adjusting the system parameters (in effect changing the Kramers rate), we can optimize the response to a given modulation frequency and background noise. In a practical system, one can move the location and height of the bell-shaped response curve of figure 1 by changing the potential parameters and, possibly, infusing noise into the system. The noise-enhancement of the SNR improves with decreasing frequency. This is a hallmark of stochastic resonance and provides one with a possible filtering technique at low frequency. It is important to point out that all the effects reported in this work have been reproduced via analog simulations (Bulsara, Jacobs, Zhou, Moss and Kiss, 1991: Zhou and Moss, 1990). Recently a new approach to the processing of information in noisy nonlinear dynamic systems, based on the probability density of residence times in one of the stable states of the potential, has been developed by Zhou, Moss and Jung (1990). This technique, which offers an alternative to the FFT, was applied by Longtin, Moss and Bulsara (1991) in their construction of the inter-spike-interval histograms that describe neuronal spike trains in the central nervous system. Their work points to the important role played by noise in the proceing of information by the central nervous system. The beneficial role of noise has already been recognized by Buhmann and Schulten (1986, 87). They found that noise, deliberately added to the deterministic equations governing individual neurons in a network significantly enhanced the network's performance and concluded that *"...the noise...is an essential feature of the information processing abilities of the neural network and not a mere source of disturbance better suppressed..."*

## Acknowledgements

This work was carried out under funding from the Office of Naval Research grant nos. N00014-90-AF-00001 and N000014-90-J-1327.

## References

Aihara K., Takake T., and Toyoda M., 1990; "Chaotic Neural Networks", Phys. Lett. **A144**, 333-340.

Buhmann J., and Schulten K., 1986; "Influence of Noise on the Behavior of an Autoassociative Neural Network", in J. Denker (ed) Neural networks for Computing (AIP conf. procedings, vol 151).

Buhmann J., and Schulten K., 1987; "Influence of Noise on the Function of a "Physiological" Neural Network", Biol. Cyber. **56**, 313-327.

Bulsara A., Boss R. and Jacobs E., 1989; "Noise Effects in an Electronic Model of a Single Neuron", Biol. Cyber. **61**, 212-222.

Bulsara A., Jacobs E., Zhou T., Moss F. and Kiss L., 1991; "Stochastic Resonance in a Single Neuron Model: Theory and Analog Simulation", J. Theor. Biol. **154**, 531-555.

Bulsara A. and Schieve W., 1991; "Single Effective Neuron: Macroscopic Potential and Noise-Induced Bifurcations", Phys. Rev. A, in press.

Englund J., Snapp R., Schieve W., 1984; "Fluctuations, Instabilities and Chaos in the Laser-Driven Nonlinear Ring Cavity", in E. Wolf (ed) Progress in Optics, vol XXI. (North Holland, Amsterdam).

Gammaitoni L., Martinelli M., Pardi L., and Santucci S., 1991; "Observation of Stochastic Resonance in Bistable Electron Paramagnetic Resonance Systems", preprint.

Hopfield J., 1982; "Neural Networks and Physical Systems with Emergent Computational Capabilities", Proc. Natl. Acad. Sci. 79, 2554-2558.

Hopfield J., 1984; "Neurons with Graded Responses have Collective Computational Abilities like those of Two-State Neurons", Proc. Natl. Acad. Sci., 81, 3088-3092.

Horsthemke W., and Lefever R., 1984; Noise-Induced Transitions. (Springer-Verlag, Berlin).

Kramers H., 1940; "Brownian Motion in a Field of Force and the Diffusion Model of Chemical Reactions", Physica 7, 284-304.

Li Z. , and Hopfield J., 1989; "Modeling the Olfactory Bulb and its Neural Oscillatoy Processings", Biol. Cyber. 61, 379-392.

Longtin A., Bulsara A., and Moss F., 1991; "Time-Interval Sequences in Bistable Systems and the Noise-Induced Transmission of Information by Sensory Neurons", Phys. Rev. Lett. 67, 656-659.

McNamara B., Wiesenfeld K., and Roy R., 1988; "Observation of Stochastic Resonance in a Ring Laser", Phys. Rev. Lett. 60, 2626-2629.

McNamara B., and Wiesenfeld K., 1989; "Theory of Stochastic Resonance", Phys. Rev. A39, 4854-4869.

Paulus M., Gass S., and Mandell A., 1990; "A Realistic Middle-Layer for Neural Networks", Physica D40, 135-155.

Rinzel J., and Ermentrout B., 1989; "Analysis of Neural Excitability and Oscillations", in Methods in Neuronal Modeling, eds. C. Koch and I. Segev (MIT Press, Cambridge, MA).

Rose J., Brugge J., Anderson D., and Hind J., 1967; "Phase-locked Response to Low-frequency Tones in Single Auditory Nerve Fibers of the Squirrel Monkey", J. Neurophysiol., 30, 769-793.

Schieve W., Bulsara A. and Davis G., 1990; "Single Effective Neuron", Phys. Rev. A43 2613-2623.

Shamma S., 1989; "Spatial and Temporal Processing in Central Auditory Networks", in Methods in Neuronal Modeling, eds. C. Koch and I. Segev (MIT Press, Cambridge, MA).

Siegal R.,1990; "Nonlinear Dynamical System Theory and Primary Visual Cortical Processing", Physica 42D, 385-395.

Spano M., and Ditto W., 1991; "Experimental Observation of Stochastic R Resonance in a Magnetoelastic Ribbon", preprint.

Tuckwell H., 1989; "Stochastic Processes in the Neurosciences", (SIAM, Philadelphia).

Vemuri G., and Roy R., 1990; "Stochastic Resonance in a Bistable Ring Laser", Phys. Rev. A39, 4668-4674.

Zhou T., and Moss F., 1990; "Analog Simulations of Stochastic Resonance", Phys. Rev. A41, 4255-4264.

Zhou T., Moss F., and Jung P., 1991; "Escape-Time Distributions of a Periodically Modulated Bistable System with Noise", Phys. Rev. A42, 3161-3169.